# Shaping the State Space Landscape in Recurrent Networks

**Patrice Y. Simard** *
Computer Science Dept.
University of Rochester
Rochester, NY 14627

**Jean Pierre Raysz**
LIUC
Université de Caen
14032 Caen Cedex
France

**Bernard Victorri**
ELSAP
Université de Caen
14032 Caen Cedex
France

## Abstract

Fully recurrent (asymmetrical) networks can be thought of as dynamic systems. The dynamics can be shaped to perform content addressable memories, recognize sequences, or generate trajectories. Unfortunately several problems can arise: First, the convergence in the state space is not guaranteed. Second, the learned fixed points or trajectories are not necessarily stable. Finally, there might exist spurious fixed points and/or spurious "attracting" trajectories that do not correspond to any patterns. In this paper, we introduce a new energy function that presents solutions to all of these problems. We present an efficient gradient descent algorithm which directly acts on the stability of the fixed points and trajectories and on the size and shape of the corresponding basin and valley of attraction. The results are illustrated by the simulation of a small content addressable memory.

## 1  INTRODUCTION

Recurrent neural networks have the capability of storing information in the state of their units. The temporal evolution of these states constitutes the dynamics of the system and depends on the weights and the input of the network. In the case of symmetric connections, the dynamics have been shown to be convergent [2] and various procedures are known for finding the weights to compute different tasks. In unconstrained neural networks however, little is known about how to train the weights of the network when the convergence of the dynamics is not guaranteed. In his review paper [1], Hirsh defines the conditions which must be satisfied for

some given dynamics to converge, but does not provide mechanisms for finding the weights to implement these dynamics.

In this paper, a new energy function is introduced which reflects the convergence and the stability of the dynamics of a network. A gradient descent procedure on the weights provides an algorithm to control interesting properties of the dynamics including contraction over a subspace, stability, and convergence.

## 2    AN ENERGY FUNCTION TO ENFORCE STABILITY

This section introduces a new energy function which can be used in combination with the backpropagation algorithm for recurrent networks (cf. [6; 5]). The continuous propagation rule is given by the equation:

$$T_i \frac{\partial x_i}{\partial t} = -x_i + g_i \left( \sum_j w_{ij} x_j \right) + I_i \tag{1}$$

where $x_i^t$ is the activation of unit $i$, $g_i$ is a differentiable function, $w_{ij}$ is the weight from unit $j$ to unit $i$, and $T_i$ and $I_i$ are respectively the time constant and the input for unit $i$. A possible discretization of this equation is

$$\tilde{x}_i^{t+1} = \tilde{x}_i^t + \frac{dt}{T_i} \left( -\tilde{x}_i + g \left( \sum_j w_{ij} \tilde{x}_j \right) + I_i \right) \tag{2}$$

$$= G_i(\tilde{x}_1^t, \tilde{x}_2^t, ..., \tilde{x}_n^t) \tag{3}$$

Where $\tilde{x}_i^t$ is the activation of unit $i$ at the discrete time step $t$. Henceforth, only the discrete version of the propagation equation will be considered and the tilda in $\tilde{x}$ will be omitted to avoid heavy notations.

### 2.1    MAKING THE MAPPING CONTRACTING OR EXPANDING IN A GIVEN DIRECTION

Using the Taylor expansion, $G(x^t + dx^t)$ can be written as

$$G(x^t + dx^t) = G(x^t) + G'(x^t) \cdot dx^t + o(\|x^t\|) \tag{4}$$

where $G'(x^t)$ is the linear application derived from $G(x^t)$ and the term $o(\|x^t\|)$ tends toward 0 faster than $\|x^t\|$. The mapping $G$ is contracting in the direction of the unitary vector $D$ if

$$
\begin{aligned}
\|G(x^t + \epsilon D) - G(x^t)\| &\leq \|\epsilon D\| \\
\epsilon \|G'(x^t) \cdot D\| &\leq \epsilon \\
\|G'(x^t) \cdot D\| &\leq 1
\end{aligned}
\tag{5}
$$

where $\epsilon$ is a small positive constant.

Accordingly, the following energy function is considered

$$E_s(X, D) = \frac{1}{2}(\|G'(X) \cdot D\|^2 - K_X)^2 \tag{6}$$

where $K_X$ is the target contracting rate at $X$ in the direction $D$. Depending on whether we choose $K_X$ larger or smaller than 1, minimizing $E_s(X, D)$ will make the mapping at $X$ contracting or expanding in the direction $D$. Note that $D$ can be a complex vector.

The variation of $E_s(X, D)$ in respect to $w_{mn}$ is equal to:

$$\frac{\partial E_s(X, D)}{\partial w_{mn}} = 2(\|G'(X)D\|^2 - K_X)\sum_i \left(\sum_j \frac{\partial G_i(X)}{\partial x_j}D_j\right)\left(\sum_j \frac{\partial^2 G_i(X)}{\partial w_{mn}\partial x_j}D_j\right) \tag{7}$$

Assuming the activation function is of the form 2, the gradient operator yields:

$$\frac{\partial G_i(X)}{\partial x_j} = \delta_{ij}(1 - \frac{dt}{T_i}) + \frac{dt}{T_i}g'(u_i)w_{ij} \tag{8}$$

where $u_i = \sum_k w_{ik}x_k$. To compute $\frac{\partial E(X,D)}{\partial w_{mn}}$ the following expression needs to be evaluated:

$$\frac{\partial^2 G_i(X)}{\partial w_{mn}\partial x_j} = \frac{dt}{T_i}\left(g''(u_i)\sum_k(\delta_{im}\delta_{kn}x_k + w_{ik}\frac{\partial x_k}{\partial w_{mn}})w_{ij} + \delta_{im}\delta_{jn}g'(u_i)\right) \tag{9}$$

which in turn requires the evaluation of $\frac{\partial x_k}{\partial w_{mn}}$. If we assume that for output units, $x_k = X_k$ and $\frac{\partial x_k}{\partial w_{mn}} = 0$, we will improve the stability of the fixed point when the visible units are clamped to the input. What we want however, is to increase stability for the network when the input units are unclamped (or hidden). This means that for every unit (including output units), we have to evaluate $\frac{\partial x_k}{\partial w_{mn}}$. Since we are at the (unstable) fixed point, we have:

$$x_i = g\left(\sum_k w_{ik}x_k\right) + I_i \tag{10}$$

If we derived this equation with respect to $w_{mn}$ we get:

$$\frac{\partial x_i}{\partial w_{mn}} = g'\left(\sum_j w_{ij}x_j\right)(\delta_{mi}x_n + \sum_j w_{ij}\frac{\partial x_j}{\partial w_{mn}}) \tag{11}$$

In matrix form:

$$c = g'(y + wc) \tag{12}$$

Where $c_i = \frac{\partial x_i}{\partial w_{mn}}$, $g'$ is a diagonal (square) matrix such that $g'_{ii} = g'_i\left(\sum_j w_{ij}x_j\right)$ and $g'_{ij} = 0$ for $i \neq j$ (note that $g'w \neq wg'$), $y$ is a vector such that $y_i = \delta_{mi}x_n$ and $w$ is the weight matrix. If we solve this we get:

$$c = (Id - g'w)^{-1}g'y \tag{13}$$

That is:

$$\frac{\partial x_i}{\partial w_{mn}} = (L^{-1})_{im}g'(u_m)x_n \tag{14}$$

where the matrix $L$ is given by:

$$L_{ij} = \delta_{ij} - g'\left(\sum_k w_{ik}x_k\right)w_{ij} \tag{15}$$

$x_k$ is the activation of the unit at the fixed point so it is the clamped value for the visible unit, and $x_k^\infty$ for the hidden unit (the system converges to a stable fixed point when the visible units are clamped).

To obtain the target rate of contraction $K_X$ at $X$ in the direction $D$, the weights are updated iteratively according to the delta rule:

$$\Delta w_{ij} = -\eta\frac{\partial E_s(X,L)}{\partial w_{ij}} \tag{16}$$

This updating rule has the advantages and disadvantages of gradient descent algorithms.

## 2.2   COMPLEXITY

The algorithm given above can be implemented in $O(N^2)$ storage and $O(N^3)$ steps, where $N$ is the number of units. This complexity however can be improved by avoiding inverting the matrix $L$ using a local algorithm such as the one presented in [7]. Another implementation of this energy function can be achieved using Lagrange multipliers. This method exactly evaluates $\frac{\partial x_i}{\partial w_{mn}}$ by using a backward pass [9]. Its complexity depends on how many steps the network is unfolded in time.

## 2.3   GLOBAL STABILITY

Global convergence can be obtained if $D$ is parallel the eigenvector corresponding to the largest eigenvalue of $G'(X)$. Indeed, in that case $G'(X) \cdot D$ is the largest eigenvalue of $G'(X)$. If $X$ is a fixed point, the Ostrowski theorem [4; 3] guarantees $X$ is stable if and only if the maximum eigenvalue of the Jacobian of $G$ is less than 1 in modulus.

Fortunately, the eigenvector corresponding to the largest eigenvalue can easily be computed using an efficient iterative method [8]. By choosing $D$ in that direction, fixed points can be made stable.

## 3   RESULTS

To simplify the following discussion, $V$ is defined to be the unitary eigenvector corresponding to the largest eigenvalue of the Jacobian of $G$.

The energy function $E_s$ can be used in at least three ways. First it can be used to accelerate the convergence toward an internal state upon presentation of a specific input $p$. This is done by increasing the rate of contraction in the direction of $V$. The chosen value for $K_{X_p}$ is therefore small with respect to 1. The resulting network will settle faster and therefore compute its output sooner. Second, $E_s$ can be used to neutralize spurious fixed points by making them unstable. If the mapping $G$

is expanding in the direction of $V$ the fixed point will be unstable, and will never be reached by the system. The corresponding target value $K_{X_p}$ should be larger than 1. Third, and most importantly, it can be used to force stability of the fixed points when doing associative memory. Recurrent backpropagation (RBP) [7] can be used to make the patterns fixed points, but there is no guarantee that these will be stable. By making $G$ contract along the direction $V$, this problem can be solved. Furthermore, one can hope that by making the eigenvalue close to 1, smoothness in the derivatives will make the basins of attraction larger. This can be used to absorb and suppress spurious neighboring stable fixed points.

The following experiment illustrates how the unstable fixed points learned with RBP can be made more stable using the energy function $E_s$. Consider a network of eight fully connected units with two visible input/output units. The network is subject to the dynamic specified by the equation 2. Three patterns are presented on the two visible units. They correspond to the coordinates of the three points $(0.3, 0.7)$, $(0.8, 0.4)$ and $(0.2, 0.1)$ which were chosen randomly. The learning phase for each pattern consists of 1) clamping the visible units while propagating for five iterations (to let the hidden units settle), 2) evaluating the difference between the activation resulting from the incoming connections of the visible units and the value of the presented pattern (this is the error), 3) backpropagating the corresponding error signals and 4) updating the weight. This procedure can be used to make a pattern a fixed point of the system [6]. Unfortunately, there is no guarantee that these fixed points will be stable. Indeed, after learning with RBP only, the maximum eigenvalue of the Jacobian of $G$ for each fixed point is shown in table 1 (column EV, no $E_s$). As can be seen, the maximum eigenvalue of two of the three patterns is larger than one.

| | unit 0 | unit 1 | EV, no $E_s$ | EV, using $E_s$ |
|---|---|---|---|---|
| pattern 0 | 0.30 | 0.70 | 1.063 | 0.966 |
| pattern 1 | 0.80 | 0.40 | 1.172 | 0.999 |
| pattern 2 | 0.20 | 0.10 | 0.783 | 0.710 |

Table 1: Patterns and corresponding norms of maximum eigenvalues (EV) of the free system, with and without the stability constraint.

For a better understanding of what this means, the network can be viewed as a dynamic system of 8 units. A projection of the dynamics of the system on the visible units can be obtained by clamping these units while propagating for five iterations, and computing the activation resulting from the incoming connection. The difference between the latter value and the pattern value is a displacement (or speed) indicating in which direction in the state space the activations are going. The corresponding vector field is plotted on the top figure 1. It can easily be seen that as predicted by the eigenvalues, patterns 0 and 1 are unstable (pattern 1 is at a saddle point) and pattern 2 is stable. Furthermore there are two additional spurious fixed points around $(0.83, 0.87)$ and $(0.78, 0.21)$.

The energy function $E_s$ can be combined with $RBP$ using the following procedure: 1) propagate a few epochs until the error is below a certain threshold $(10^{-5})$, 2) for each pattern, estimate the largest eigenvalue $\lambda$ and the corresponding eigenvector $V$ and 3), update the weights using $E_s$ until $|\lambda| < K$ in direction $V$. Steps 1 to 3

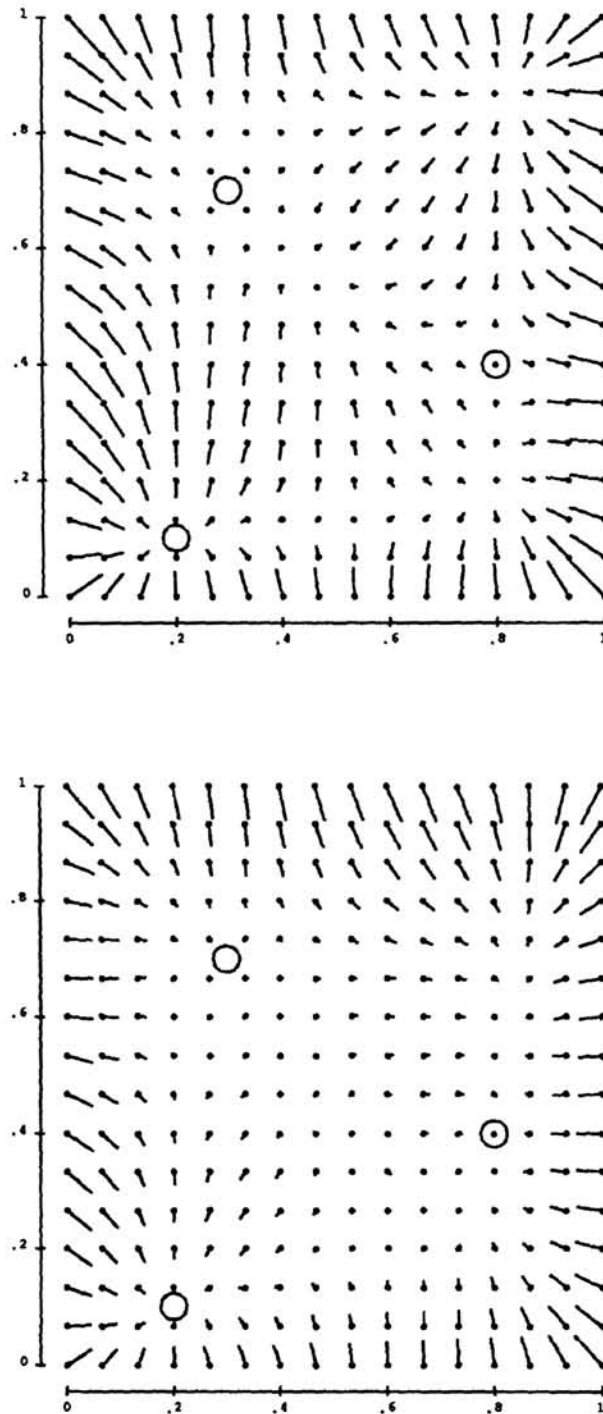

Figure 1: Vector fields representing the dynamics of the state space after learning the patterns $(0.3, 0.7)$, $(0.8, 0.4)$ and $(0.2, 0.1)$. The field on the top represents the dynamics of the network after training with the standard backpropagation algorithm. The field on the bottom represents the dynamics of the network after training with the standard backpropagation algorithm combined with $E_s$.

are repeated until no more progress is made. The largest eigenvalues after learning are shown in table 1 in the last column. As can be noticed, all the eigenvalues are less than one and therefore the mapping $G$ is contracting in all directions. The dynamics of the network is plotted at the bottom of figure 1. As can clearly be seen, all the patterns are now attractors. Furthermore the two spurious fixed points have disappeared in the large basin of attraction of pattern 1. This is a net improvement over RBP used alone, since the network can now be used as a content addressable memory.

## 4  DISCUSSION

In this paper we have introduced mechanisms to control global aspects such as stability, attractor size, or contraction speed, of the dynamics of a recurrent network. The power of the algorithm is illustrated by implementing a content addressable memory with an asymmetric neural network. After learning, the stable fixed points of the system coincide with the target patterns. All spurious fixed points have been eliminated by spreading the basins of attraction of the target patterns.

The main limitation of the algorithm resides in using a gradient descent to update the weights. Parameters such as the learning rate have to be carefully chosen, for optimal performance. Furthermore, there is always a possibility that the evolution of the weights might be trapped in a local minimum.

The complexity of the algorithm can be further improved. In equation 10 for instance, it is assumed that we are at a fixed point. This assumption is not true unless the RBP error is really small. This requires that the RBP and the $E_s$ algorithms are run alternatively. A faster and more robust method consists in using backpropagation in time to compute $\frac{\partial x}{\partial w_{mn}}$ and is presently under study.

Finally, the algorithm can be generalized to control the dynamics around target trajectories, such as in [5]. The dynamics is projected onto the hyperplane orthogonal to the state space trajectory and constraints can be applied on the projected dynamics.

### Acknowledgements

This material is based upon work supported by the National Science Foundation under Grant number IRI-8903582.

We thank the S.H.S department of C.N.R.S (France), Neuristic Inc. for allowing the use of its neural net simulator SN2, and Corinna Cortes for helpful comments and support.

## Footnotes

*Now with AT&T Bell Laboratories, Crawfords Corner Road, Holmdel, NJ 07733

## References

[1] Morris W. Hirsch. Convergent activation dynamics in continuous time networks. *Neural Networks*, 2:331–349, 1989.

[2] J. J. Hopfield. Neural networks and physical systems with emergent collective computational abilities. *Proceedings of the National Academy of Sciences*, 79:2554–2558, April 1982.

[3] J. Ortega and M. Rockoff. Nonlinear difference equations and gauss-seidel type iterative methods. *SIAM J. Numer. Anal.*, 3:497–513, 1966.

[4] A Ostrowski. *Solutions of Equations and Systems of Equations*. Academic Press, New York, 1960.

[5] Barak Pearlmutter. Learning state space trajectories in recurrent neural networks. *Neural Computation*, 1(2):263–269, 1989.

[6] Fernado J. Pineda. Dynamics and architecture in neural computation. *Journal of Complexity*, 4:216–245, 1988.

[7] Fernando J. Pineda. Generalization of backpropagation to recurrent and higher order networks. In *IEEE Conference on Neural Information Processing Systems*, pages 602–601. American Institute of Physics, 1987.

[8] Anthony Ralston and Philip Rabinowitz. *A First Course in Numerical Analysis*. McGraw-Hill, New York, 1978.

[9] Patrice Y. Simard. *Learning State Space Dynamics in Recurrent Networks*. PhD thesis, University of Rochester, 1991.
